# Learning person-object interactions for action recognition in still images

**Vincent Delaitre**[*]
École Normale Supérieure

**Josef Sivic**[*]
INRIA Paris - Rocquencourt

**Ivan Laptev**[*]
INRIA Paris - Rocquencourt

## Abstract

We investigate a discriminatively trained model of person-object interactions for recognizing common human actions in still images. We build on the locally order-less spatial pyramid bag-of-features model, which was shown to perform extremely well on a range of object, scene and human action recognition tasks. We introduce three principal contributions. First, we replace the standard quantized local HOG/SIFT features with stronger discriminatively trained body part and object detectors. Second, we introduce new person-object interaction features based on spatial co-occurrences of individual body parts and objects. Third, we address the combinatorial problem of a large number of possible interaction pairs and propose a discriminative selection procedure using a linear support vector machine (SVM) with a sparsity inducing regularizer. Learning of action-specific body part and object interactions bypasses the difficult problem of estimating the complete human body pose configuration. Benefits of the proposed model are shown on human action recognition in consumer photographs, outperforming the strong bag-of-features baseline.

## 1  Introduction

Human actions are ubiquitous and represent essential information for understanding the content of many still images such as consumer photographs, news images, sparsely sampled surveillance videos, and street-side imagery. Automatic recognition of human actions and interactions, however, remains a very challenging problem. The key difficulty stems from the fact that the imaged appearance of a person performing a particular action can vary significantly due to many factors such as camera viewpoint, person's clothing, occlusions, variation of body pose, object appearance and the layout of the scene. In addition, motion cues often used to disambiguate actions in video [6, 27, 31] are not available in still images.

In this work, we seek to recognize common human actions, such as "walking", "running" or "reading a book" in challenging realistic images. As opposed to action recognition in video [6, 27, 31], action recognition in still images has received relatively little attention. A number of previous works [21, 24, 37] focus on exploiting body pose as a cue for action recognition. In particular, several methods address joint modeling of human poses, objects and relations among them [21, 40]. Reliable estimation of body configurations for people in arbitrary poses, however, remains a very challenging research problem. Less structured representations, e.g. [11, 39] have recently emerged as a promising alternative demonstrating state-of-the-art results for action recognition in static images.

In this work, we investigate discriminatively trained models of interactions between objects and human body parts. We build on the locally orderless statistical representations based on spatial

---

[*]WILLOW project, Laboratoire d'Informatique de l'École Normale Supérieure, ENS/INRIA/CNRS UMR 8548, Paris, France

pyramids [28] and bag-of-features models [9, 16, 34], which have demonstrated excellent performance on a range of scene [28], object [22, 36, 41] and action [11] recognition tasks. Rather than relying on accurate estimation of body part configurations or accurate object detection in the image, we represent human actions as locally orderless distributions over body parts and objects together with their interactions. By opportunistically learning class-specific object and body part interactions (e.g. relative configuration of leg and horse detections for the riding horse action, see Figure 1), we avoid the extremely challenging task of estimating the full body configuration. Towards this goal, we consider the following challenges: (i) what should be the representation of object and body part appearance; (ii) how to model object and human body part interactions; and (iii) how to choose suitable interaction pairs in the huge space of all possible combinations and relative configurations of objects and body parts.

To address these challenges, we introduce the following three contributions. First, we replace the quantized HOG/SIFT features, typically used in bag-of-features models [11, 28, 36] with powerful, discriminatively trained, local object and human body part detectors [7, 25]. This significantly enhances generalization over appearance variation, due to e.g. clothing or viewpoint while providing a reliable signal on part locations. Second, we develop a part interaction representation, capturing pair-wise relative position and scale between object/body parts, and include this representation in a scale-space spatial pyramid model. Third, rather than choosing interacting parts manually, we select them in a discriminative fashion. Suitable pair-wise interactions are first chosen from a large pool of hundreds of thousands of candidate interactions using a linear support vector machine (SVM) with a sparsity inducing regularizer. The selected interaction features are then input into a final, more computationally expensive, non-linear SVM classifier based on the locally orderless spatial pyramid representation.

## 2 Related work

Modeling person-object interactions for action recognition has recently attracted significant attention. Gupta *et al.* [21], Wang *et al.* [37], and Yao and Fei Fei [40] develop joint models of body pose configuration and object location within the image. While great progress has been made on estimating body pose configurations [5, 19, 25, 33], inferring accurate human body pose in images of common actions in consumer photographs remains an extremely challenging problem due to a significant amount of occlusions, partial truncation by image boundaries or objects in the scene, non-upright poses, and large variability in camera viewpoint.

While we build on the recent body pose estimation work by using strong pose-specific body part models [7, 25], we explicitly avoid inferring the complete body configuration. In a similar spirit, Desai *et al.* [13] avoid inferring body configuration by representing a small set of body postures using single HOG templates and represent relative position of the entire person and an object using simple relations (e.g. above, to the left). They do not explicitly model body parts and their interactions with objects as we do in this work. Yang *et al.* [38] model the body pose as a latent variable for action recognition. Differently to our method, however, they do not attempt to model interactions between people (their body parts) and objects. In a recent work, Maji *et al.* [30] also represent people by activation responses of body part detectors (rather than inferring the actual body pose), however, they model only interactions between person and object bounding boxes, not considering individual body parts, as we do in this work.

Learning spatial groupings of low-level (SIFT) features for recognizing person-object interactions has been explored by Yao and Fei Fei [39]. While we also learn spatial interactions, we build on powerful body part and object detectors pre-learnt on separate training data, providing a degree of generalization over appearance (e.g. clothing), viewpoint and illumination variation. Differently to [39], we deploy dicriminative selection of interactions using SVM with sparsity inducing regularizer.

Spatial-pyramid based bag-of-features models have demonstrated excellent performance on action recognition in still images [1, 11] outperforming body pose based methods [21] or grouplet models [40] on their datasets [11]. We build on these locally orderless representations but replace the low-level features (HOG) with strong pre-trained detectors. Similarly, the object-bank representation [29], where natural scenes are represented by response vectors of densely applied pre-trained

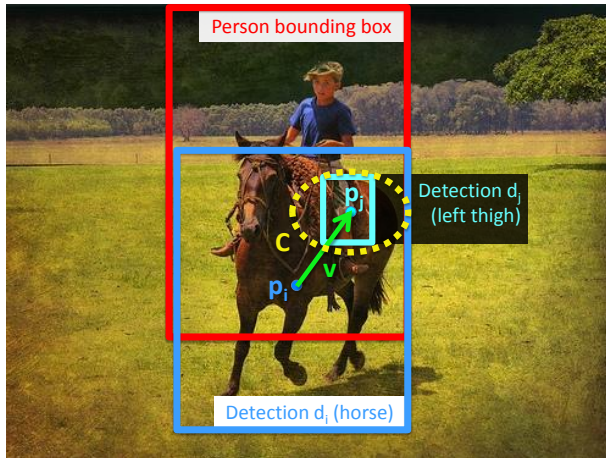

Figure 1: **Representing person-object interactions** by pairs of body part (cyan) and object (blue) detectors. To get a strong interaction response, the pair of detectors (here visualized at positions $\mathbf{p_i}$ and $\mathbf{p_j}$) must fire in a particular relative 3D scale-space displacement (given by the vector $\mathbf{v}$) with a scale-space displacement uncertainty (deformation cost) given by diagonal $3{\times}3$ covariance matrix $\mathbf{C}$ (the spatial part of $\mathbf{C}$ is visualized as a yellow dotted ellipse). Our image representation is defined by the max-pooling of interaction responses over the whole image, solved efficiently by the distance transform.

object detectors, has shown a great promise for scene recognition. The work in [29], however, does not attempt to model people, body parts and their interactions with objects.

Related work also includes models of contextual spatial and co-occurrence relationships between objects [12, 32] as well as objects and the scene [22, 23, 35]. Object part detectors trained from labelled data also form a key ingredient of attribute-based object representations [15, 26]. While we build on this body of work, these approaches do not model interactions of people and their body parts with objects and focus on object/scene recognition rather than recognition of human actions.

## 3 Representing person-object interactions

This section describes our image representation in terms of body parts, objects and interactions among them.

### 3.1 Representing body parts and objects

We assume to have a set of $n$ available detectors $d_1, \ldots, d_n$ which have been pre-trained for different body parts and object classes. Each detector $i$ produces a map of dense 3D responses $d_i(\mathbf{I}, \mathbf{p})$ over locations and scales of a given image $\mathbf{I}$. We express the positions of detections $\mathbf{p}$ in terms of scale-space coordinates $\mathbf{p} = (x, y, \sigma)$ where $(x, y)$ corresponds to the spatial location and $\sigma = \log \tilde{\sigma}$ is an additive scale parameter log-related to the image scale factor $\tilde{\sigma}$ making the addition in the position vector space meaningful.

In this paper we use two types of detectors. For objects we use LSVM detector [17] trained on PASCAL VOC images for ten object classes[1]. For body parts we implement the method of [25] and train ten body part detectors[2] for each of sixteen pose clusters giving 160 body part detectors in total (see [25] for further details). Both of our detectors use Histograms of Oriented Gradients (HOG) [10] as an underlying low-level image representation.

## 3.2 Representing pairwise interactions

We define interactions by the pairs of detectors $(d_i, d_j)$ as well as by the spatial and scale relations among them. Each pair of detectors constitutes a two-node tree where the position and the scale of the leaf are related to the root by scale-space offset and a spatial deformation cost. More precisely, an interaction pair is defined by a quadruplet $\mathbf{q} = (i, j, \mathbf{v}, \mathbf{C}) \in \mathbb{N} \times \mathbb{N} \times \mathbb{R}^3 \times \mathbb{M}_{3,3}$ where $i$ and $j$ are the indices of the detectors at the root and leaf, $\mathbf{v}$ is the offset of the leaf relatively to the root and $\mathbf{C}$ is a $3 \times 3$ diagonal matrix defining the displacement cost of the leaf with respect to its expected position. Figure 1 illustrates an example of an interaction between a horse and the left thigh for the horse riding action.

We measure the response of the interaction $\mathbf{q}$ located at the root position $\mathbf{p_1}$ by:

$$r(\mathbf{I}, \mathbf{q}, \mathbf{p_1}) = \max_{\mathbf{p_2}} \left( d_i(\mathbf{I}, \mathbf{p_1}) + d_j(\mathbf{I}, \mathbf{p_2}) - \mathbf{u}^T \mathbf{C} \mathbf{u} \right) \tag{1}$$

where $\mathbf{u} = \mathbf{p_2} - (\mathbf{p_1} + \mathbf{v})$ is the displacement vector corresponding to the drift of the leaf node with respect to its expected position $(\mathbf{p_1} + \mathbf{v})$. Maximizing over $\mathbf{p_2}$ in (1) provides localization of the leaf node with the optimal trade-off between the detector score and the displacement cost. For any interaction $\mathbf{q}$ we compute its responses for all pairs of node positions $\mathbf{p_1}, \mathbf{p_2}$. We do this efficiently in linear time with respect to $\mathbf{p}$ using distance transform [18].

## 3.3 Representing images by response vectors of pair-wise interactions

Given a set of $M$ interaction pairs $\mathbf{q}_1, \cdots, \mathbf{q}_M$, we wish to aggregate their responses (1), over an image region $\mathcal{A}$. Here $\mathcal{A}$ can be (i) an (extended) person bounding box, as used for selecting discriminative interaction features (Section 4.2) or (ii) a cell of the scale-space pyramid representation, as used in the final non-linear classifier (Section 4.3). We define score $s(\mathbf{I}, \mathbf{q}, \mathcal{A})$ of an interaction pair $\mathbf{q}$ within $\mathcal{A}$ of an image $\mathbf{I}$ by max-pooling, i.e. as the maximum response of the interaction pair within $\mathcal{A}$:

$$s(\mathbf{I}, \mathbf{q}, \mathcal{A}) = \max_{\mathbf{p} \in \mathcal{A}} r(\mathbf{I}, \mathbf{q}, \mathbf{p}). \tag{2}$$

An image region $\mathcal{A}$ is then represented by a $M$-vector of interaction pair scores

$$\mathbf{z} = (s_1, \cdots, s_M) \text{ with } s_i = s(\mathbf{I}, \mathbf{q}_i, \mathcal{A}). \tag{3}$$

# 4 Learning person-object interactions

Given object and body part interaction pairs $\mathbf{q}$ introduced in the previous section, we wish to use them for action classification in still images. A brute-force approach of analyzing all possible interactions, however, is computationally prohibitive since the space of all possible interactions is combinatorial in the number of detectors and scale-space relations among them. To address this problem, we aim in this paper to select a set of $M$ action-specific interaction pairs $\mathbf{q}_1, \ldots, \mathbf{q}_M$, which are both representative and discriminative for a given action class. Our learning procedure consists of the three main steps as follows. First, for each action we generate a large pool of candidate interactions, each comprising a pair of (body part / object) detectors and their relative scale-space displacement. This step is data-driven and selects candidate detection pairs which frequently occur for a particular action in a consistent relative scale-space configuration. Next, from this initial pool of candidate interactions we select a set of $M$ discriminative interactions which best separate the particular action class from other classes in our training set. This is achieved using a linear Support Vector Machine (SVM) classifier with a sparsity inducing regularizer. Finally, the discriminative interactions are combined across classes and used as interaction features in our final non-linear spatial-pyramid like SVM classifier. The three steps are detailed below.

## 4.1 Generating a candidate pool of interaction pairs

To initialize our model, we first generate a large pool of candidate interactions in a data-driven manner. Following the suggestion in [17] that the accurate selection of the deformation cost $\mathbf{C}$ may not be that important, we set $\mathbf{C}$ to a reasonable fixed value for all pairs, and focus on finding clusters of frequently co-occurring detectors $(d_i, d_j)$ in specific relative configurations.

For each detector $i$ and an image $\mathbf{I}$, we first collect a set of positions of all positive detector responses $\mathbf{P_i^I} = \{\mathbf{p} \mid d_i(\mathbf{I}, \mathbf{p}) > 0\}$, where $d_i(\mathbf{I}, \mathbf{p})$ is the response of detector $i$ at position $\mathbf{p}$ in image $\mathbf{I}$. We

then apply a standard non-maxima suppression (NMS) step to eliminate multiple responses of a detector in local image neighbourhoods and then limit $\mathbf{P}_i^{\mathbf{I}}$ to the $L$ top-scoring detections. The intuition behind this step is that a part/object interaction is not likely to occur many times in an image.

For each pair of detectors $(d_i, d_j)$ we then gather relative displacements between their detections from all the training images $\mathbf{I}_k$: $\mathbf{D}_{ij} = \bigcup_k \{ p_j - p_i \mid p_i \in \mathbf{P}_i^{\mathbf{I}_k} \text{ and } p_j \in \mathbf{P}_j^{\mathbf{I}_k} \}$. To discover potentially interesting interaction pairs, we perform a mean-shift clustering over $\mathbf{D}_{ij}$ using a window of radius $\mathbf{R} \in \mathbb{R}_3$ (2D-image space and scale) equal to the inverse of the square root of the deformation cost: $\mathbf{R} = diag(\mathbf{C}^{-\frac{1}{2}})$. We also discard clusters which contribute to less than $\eta$ percent of the training images. The set of $m$ resulting candidate pairs $(i, j, \mathbf{v}_1, \mathbf{C}), \cdots, (i, j, \mathbf{v}_m, \mathbf{C})$ is built from the centers $\mathbf{v}_1, \cdots, \mathbf{v}_m$ of the remaining clusters. By applying this procedure to all pairs of detectors, we generate a large pool (hundreds of thousands) of potentially interesting candidate interactions.

## 4.2 Discriminative selection of interaction pairs

The initialization described above produces a large number of candidate interactions. Many of them, however, may not be informative resulting in unnecessary computational load at the training and classification times. For this reason we wish to select a smaller number of $M$ discriminative interactions.

Given a set of $N$ training images, each represented by an interaction response vector $\mathbf{z_i}$, described in eq. (3) where $\mathcal{A}$ is the extended person bounding box given for each image, and a binary label $y_i$ (in a 1-vs-all setup for each class), the learning problem for each action class can be formulated using the binary SVM cost function:

$$J(\mathbf{w}, b) = \lambda \sum_{i=1}^{N} \max\{0, 1 - y_i(\mathbf{w}^\top \mathbf{z_i} + b)\} + \|\mathbf{w}\|_1, \qquad (4)$$

where $\mathbf{w}, b$ are parameters of the classifier and $\lambda$ is the weighting factor between the (hinge) loss on the training examples and the $L_1$ regularizer of the classifier.

By minimizing (4) in a one-versus-all setting for each action class we search (by binary search) for the value of the regularization parameter $\lambda$ resulting in the sparse weight vector $\mathbf{w}$ with $M$ non-zero elements. Selection of $M$ interaction pairs corresponding to non-zero elements of $\mathbf{w}$ gives $M$ most discriminative (according to (4)) interaction pairs per action class. Note that other discriminative feature selection strategies such as boosting [20] can be also used. However, the proposed approach is able to jointly search the entire set of candidate feature pairs by minimizing a convex cost given in (4), whereas boosting implements a greedy feature selection procedure, which may be sub-optimal.

## 4.3 Using interaction pairs for classification

Given a set of $M$ discriminative interactions for each action class obtained as described above, we wish to train a final non-linear action classifier. We use spatial pyramid-like representation [28], aggregating responses in each cell of the pyramid using max-pooling as described by eq. (2), where $\mathcal{A}$ is one cell of the spatial pyramid. We extend the standard 2D pyramid representation to scale-space resulting in a 3D pyramid with $D = 1 + 2^3 + 4^3 = 73$ cells. Using the scale-space pyramid with $D$ cells, we represent each image by concatenating $M$ features from each of the $K$ classes into a $MKD$-dimensional vector. We train a non-linear SVM with RBF kernel and $L_2$ regularizer for each action class using a 5-fold cross-validation for the regularization and kernel band-width parameters. We found that using this final non-linear classifier consistently improves classification performance over the linear SVM given by equation (4). Note that feature selection (section 4.2) is necessary in this case as applying the non-linear spatial pyramid classifier on the entire pool of all candidate interactions would be computationally infeasible.

## 5 Experiments

We test our model on the Willow-action dataset downloaded from [4] and the PASCAL VOC 2010 action classification dataset [14]. The Willow-action dataset contains more than 900 images with more than 1100 labelled person detections from 7 human action classes: *Interaction with Computer,*

*Photographing, Playing Music, Riding Bike, Riding Horse, Running* and *Walking*. The training set contains 70 examples of each action class and the rest (at least 39 examples per class) is left for testing. The PASCAL VOC 2010 dataset contains the 7 above classes together with 2 other actions: *Phoning* and *Reading*. It contains a similar number of images. Each training and testing image in both datasets is annotated with the smallest bounding box containing each person and by the performed action(s). We follow the same experimental setup for both datasets.

**Implementation details:** We use our implementation of body part detectors described in [25] with 16 pose clusters trained on the publicly available 2000 image database [3], and 10 pre-trained PAS-CAL 2007 Latent SVM object detectors [2]: *bicycle, car, chair, cow, dining table, horse, motorbike, person, sofa, tvmonitor*. In the human action training/test data, we extend each given person bounding box by 50% and resize the image so that the bounding box has a maximum size of 300 pixels. We run the detectors over the transformed bounding boxes and consider the image scales $s_k = 2^{k/10}$ for $k \in \{-10, \cdots, 10\}$. At each scale we extract the detector response every 4 pixels and 8 pixels for the body part and object detectors, respectively. The outputs of each detector are then normalized by subtracting the mean of maximum responses within the training bounding boxes and then normalizing the variance to 1. We generate the candidate interaction pairs by taking the mean-shift radius $\mathbf{R} = (30, 30, \log(2)/2)$, $L = 3$ and $\eta = 8\%$. The covariance of the pair deformation cost $\mathbf{C}$ is fixed in all experiments to $\mathbf{R}^{-2}$. We select $M = 310$ discriminative interaction pairs to compute the final spatial pyramid representation of each image.

**Results:** Table 1 summarizes per-class action classification results (reported using average precision for each class) for the proposed method (d. Interactions), and three baselines. The first baseline (a. BOF) is the bag-of-features classifier [11], aggregating quantized responses of densely sampled HOG features in spatial pyramid representation, using a (non-linear) intersection kernel. Note that this is a strong baseline, which was shown [11] to outperform the recent person-object interaction models of [39] and [21] on their own datasets. The second baseline (b. LSVM) is the latent SVM classifier [17] trained in a 1-vs-all fashion for each class. To obtain a single classification score for each person bounding box, we take the maximum LSVM detection score from the detections overlapping the extended bounding box with the standard overlap score [14] higher than 0.5. The final baseline (c. Detectors) is a SVM classifier with an RBF kernel trained on max-pooled responses of the entire bank of body part and object detectors in a spatial pyramid representation but without interactions. This baseline is similar in spirit to the object bank representation [29], but here targeted to action classification by including a bank of pose-specific body part detectors as well as object detectors. On average, the proposed method (d.) outperforms all baselines, obtaining the best result on 4 out of 7 classes. The largest improvements are obtained on Riding Bike and Horse actions, for which reliable object detectors are available. The improvement of the proposed method d. with respect to using the plain bank of object and body part detectors c. directly demonstrates the benefit of modeling interactions. Example detections of interaction pairs are shown in figure 2.

Table 2 shows the performance of the proposed interaction model (d. Interactions) and its combination with the baselines (e. BOF+LSVM+Inter.) on the Pascal VOC 2010 data. Interestingly, the proposed approach is complementary to both the BOF (51.25 mAP) and LSVM (44.08 mAP) methods and by combining all three approaches (following [11]) the overall performance improves to 60.66 mAP. We also report results of the "Poselet" method [30], which, similar to our method, is trained from external non-Pascal data. Our combined approach achieves better overall performance and also outperforms the "Poselet" approach on 6 out of 9 classes. Finally, our combined approach also obtains competitive performance compared to the overall best reported result on the Pascal VOC 2010 data – "SURREY_MK_KDA" [1] – and outperforms this method on the "Riding Horse" and "Walking" classes.

## 6 Conclusion

We have developed person-object interaction features based on non-rigid relative scale-space displacement of pairs of body part and object detectors. Further, we have shown that such features can be learnt in a discriminative fashion and can improve action classification performance over a strong bag-of-features baseline in challenging realistic images of common human actions. In addition, the learnt interaction features in some cases correspond to visually meaningful configurations of body parts, and body parts with objects.

| | | | | |
|---|---|---|---|---|
| **Inter. w/ Comp.**<br>Blue: Screen<br>Cyan: L. Leg | | | | |
| **Photographing**<br>Blue: Head<br>Cyan: L. Thigh | | | | |
| **Playing Instr.**<br>Blue: L. Forearm<br>Cyan: L. Forearm | | | | |
| **Riding Bike**<br>Blue: R. Forearm<br>Cyan: Motorbike | | | | |
| **Riding Horse**<br>Blue: Horse<br>Cyan: L. Thigh | | | | |
| **Running**<br>Blue: L. Arm<br>Cyan: R. Leg | | | | |
| **Walking**<br>Blue: L. Arm<br>Cyan: Head | | | | |

Figure 2: **Example detections of discriminative interaction pairs.** These body part interaction pairs are chosen as discriminative (high positive weight $w_i$) for action classes indicated on the left. In each row, the first three images show detections on the correct action class. The last image shows a high scoring detection on an incorrect action class. In the examples shown, the interaction features capture either a body part and an object, or two body part interactions. Note that while these interaction pairs are found to be discriminative, due to the detection noise, they do not necessary localize the correct body parts in all images. However, they may still fire at consistent locations across many images as illustrated in the second row, where the head detector consistently detects the camera lens, and the thigh detector fires consistently at the edge of the head. Similarly, the leg detector seems to consistently fire on keyboards (see the third image in the first row for an example), thus improving the confidence of the computer detections for the "Interacting with computer" action.

| Action / Method | a. BOF [11] | b. LSVM | c. Detectors | d. Interactions |
|---|---|---|---|---|
| (1) Inter. w/ Comp. | **58.15** | 30.21 | 45.64 | 56.60 |
| (2) Photographing | 35.39 | 28.12 | 36.35 | **37.47** |
| (3) Playing Music | **73.19** | 56.34 | 68.35 | 72.00 |
| (4) Riding Bike | 82.43 | 68.70 | 86.69 | **90.39** |
| (5) Riding Horse | 69.60 | 60.12 | 71.44 | **75.03** |
| (6) Running | 44.53 | 51.99 | 57.65 | **59.73** |
| (7) Walking | 54.18 | 55.97 | **57.68** | 57.64 |
| **Average (mAP)** | 59.64 | 50.21 | 60.54 | **64.12** |

Table 1: Per-class average-precision for different methods on the Willow-actions dataset.

| Action / Method | d. Interactions | e. BOF+LSVM+Inter. | Poselets[30] | MK-KDA[1] |
|---|---|---|---|---|
| (1) Phoning | 42.11 | 48.61 | 49.6 | **52.6** |
| (2) Playing Instr. | 30.78 | 53.07 | 43.2 | **53.5** |
| (3) Reading | 28.70 | 28.56 | 27.7 | **35.9** |
| (4) Riding Bike | **84.93** | 80.05 | 83.7 | 81.0 |
| (5) Riding Horse | 89.61 | **90.67** | 89.4 | 89.3 |
| (6) Running | 81.28 | 85.81 | 85.6 | **86.5** |
| (7) Taking Photo | 26.89 | **33.53** | 31.0 | 32.8 |
| (8) Using Computer | 52.31 | 56.10 | 59.1 | **59.2** |
| (9) Walking | **70.12** | 69.56 | 67.9 | 68.6 |
| **Average (mAP)** | 56.30 | 60.66 | 59.7 | **62.2** |

Table 2: Per-class average-precision on the Pascal VOC 2010 action classification dataset.

We use only a small set of object detectors available at [2], however, we are now in a position to include many more additional object (camera, computer, laptop) or texture (grass, road, trees) detectors, trained from additional datasets, such as ImageNet or LabelMe. Currently, we consider detections of entire objects, but the proposed model can be easily extended to represent interactions between body parts and parts of objects [8].

**Acknowledgements.** This work was partly supported by the Quaero, OSEO, MSR-INRIA, ANR DETECT (ANR-09-JCJC-0027-01) and the EIT-ICT labs.

## Footnotes

[1]The ten object detectors correspond to object classes *bicycle, car, chair, cow, dining table, horse, motorbike, person, sofa, tv/monitor*

[2]The ten body part detectors correspond to *head, torso, {left, right} × {forearm, upper arm, lower leg, thigh}*

# References

[1] http://pascallin.ecs.soton.ac.uk/challenges/voc/voc2010/results/index.html.

[2] http://people.cs.uchicago.edu/~pff/latent/.

[3] http://www.comp.leeds.ac.uk/mat4saj/lsp.html.

[4] http://www.di.ens.fr/willow/research/stillactions/.

[5] M. Andriluka, S. Roth, and B. Schiele. Pictorial structures revisited: People detection and articulated pose estimation. In *CVPR*, 2009.

[6] A. Bobick and J. Davis. The recognition of human movement using temporal templates. *IEEE PAMI*, 23(3):257–276, 2001.

[7] L. Bourdev and J. Malik. Poselets: Body part detectors trained using 3D human pose annotations. In *ICCV*, 2009.

[8] T. Brox, L. Bourdev, S. Maji, and J. Malik. Object segmentation by alignment of poselet activations to image contours. In *CVPR*, 2011.

[9] G. Csurka, C. Bray, C. Dance, and L. Fan. Visual categorization with bags of keypoints. In *WS-SLCV, ECCV*, 2004.

[10] N. Dalal and B. Triggs. Histograms of oriented gradients for human detection. In *CVPR*, pages I:886–893, 2005.

[11] V. Delaitre, I. Laptev, and J. Sivic. Recognizing human actions in still images: a study of bag-of-features and part-based representations. In *Proc. BMVC.*, 2010. updated version, available at http://www.di.ens.fr/willow/research/stillactions/.

[12] C. Desai, D. Ramanan, and C. Fowlkes. Discriminative models for multi-class object layout. In *ICCV*, 2009.

[13] C. Desai, D. Ramanan, and C. Fowlkes. Discriminative models for static human-object interactions. In *SMiCV, CVPR*, 2010.

[14] M. Everingham, L. Van Gool, C. Williams, J. Winn, and A. Zisserman. The pascal visual object classes (voc) challenge. *IJCV*, 2010. In press.

[15] A. Farhadi, I. Endres, D. Hoiem, and D. Forsyth. Describing objects by their attributes. In *CVPR*, 2009.

[16] L. Fei-Fei and P. Perona. A Bayesian hierarchical model for learning natural scene categories. In *CVPR*, Jun 2005.

[17] P. Felzenszwalb, R. Girshick, D. McAllester, and D. Ramanan. Object detection with discriminatively trained part based models. *IEEE PAMI*, 2009.

[18] P. Felzenszwalb and D. Huttenlocher. Distance transforms of sampled functions. Technical report, Cornell University CIS, Tech. Rep. 2004-1963, 2004.

[19] V. Ferrari, M. Marin-Jimenez, and A. Zisserman. Pose search: retrieving people using their pose. In *CVPR*, 2009.

[20] Y. Freund and R. Schapire. A decision theoretic generalisation of online learning. *Computer and System Sciences*, 55(1):119–139, 1997.

[21] A. Gupta, A. Kembhavi, and L. Davis. Observing human-object interactions: Using spatial and functional compatibility for recognition. *IEEE PAMI*, 31(10):1775–1789, 2009.

[22] H. Harzallah, F. Jurie, and C. Schmid. Combining efficient object localization and image classification. In *ICCV*, 2009.

[23] D. Hoiem, A. Efros, and M. Hebert. Putting objects in perspective. In *CVPR*, 2006.

[24] N. Ikizler, R. G. Cinbis, S. Pehlivan, and P. Duygulu. Recognizing actions from still images. In *Proc. ICPR*, 2008.

[25] S. Johnson and M. Everingham. Learning effective human pose estimation from inaccurate annotation. In *CVPR*, 2011.

[26] C. Lampert, H. Nickisch, and S. Harmeling. Learning to detect unseen object classes by between-class attribute transfer. In *CVPR*, 2009.

[27] I. Laptev, M. Marszałek, C. Schmid, and B. Rozenfeld. Learning realistic human actions from movies. In *CVPR*, 2008.

[28] S. Lazebnik, C. Schmid, and J. Ponce. Beyond bags of features: spatial pyramid matching for recognizing natural scene categories. In *CVPR*, pages II: 2169–2178, 2006.

[29] L. Li, H. Su, E. Xing, and L. Fei-Fei. Object bank: A high-level image representation for scene classification and semantic feature sparsification. In *NIPS*, 2010.

[30] S. Maji, L. Bourdev, and J. Malik. Action recognition from a distributed representation of pose and appearance. In *CVPR*, 2011.

[31] T. B. Moeslund, A. Hilton, and V. Kruger. A survey of advances in vision-based human motion capture and analysis. *CVIU*, 103(2-3):90–126, 2006.

[32] A. Rabinovich, A. Vedaldi, C. Galleguillos, E. Wiewiora, and S. Belongie. Objects in context. In *ICCV*, 2007.

[33] B. Sapp, A. Toshev, and B. Taskar. Cascaded models for articulated pose estimation. In *ECCV*, 2010.

[34] J. Sivic and A. Zisserman. Video Google: A text retrieval approach to object matching in videos. In *ICCV*, 2003.

[35] A. Torralba. Contextual priming for object detection. *IJCV*, 53(2):169–191, July 2003.

[36] A. Vedaldi, V. Gulshan, M. Varma, and A. Zisserman. Multiple kernels for object detection. In *ICCV*, 2009.

[37] Y. Wang, H. Jiang, M. S. Drew, Z. N. Li, and G. Mori. Unsupervised discovery of action classes. In *CVPR*, pages II: 1654–1661, 2006.

[38] W. Yang, Y. Wang, and G. Mori. Recognizing human actions from still images with latent poses. In *CVPR*, 2010.

[39] B. Yao and L. Fei-Fei. Grouplet: A structured image representation for recognizing human and object interactions. In *CVPR*, 2010.

[40] B. Yao and L. Fei-Fei. Modeling mutual context of object and human pose in human-object interaction activities. In *CVPR*, 2010.

[41] J. Zhang, M. Marszalek, S. Lazebnik, and C. Schmid. Local features and kernels for classification of texture and object categories: a comprehensive study. *IJCV*, 73(2):213–238, 2007.

